# Analog Computation at a Critical Point: A Novel Function for Neuronal Oscillations?

**Leonid Kruglyak and William Bialek**
Department of Physics
University of California at Berkeley
Berkeley, California 94720
and NEC Research Institute*
4 Independence Way
Princeton, New Jersey 08540

## Abstract

We show that a simple spin system biased at its critical point can encode spatial characteristics of external signals, such as the dimensions of "objects" in the visual field, in the temporal correlation functions of individual spins. Qualitative arguments suggest that regularly firing neurons should be described by a planar spin of unit length, and such XY models exhibit critical dynamics over a broad range of parameters. We show how to extract these spins from spike trains and then measure the interaction Hamiltonian using simulations of small clusters of cells. Static correlations among spike trains obtained from simulations of large arrays of cells are in agreement with the predictions from these Hamiltonians, and dynamic correlations display the predicted encoding of spatial information. We suggest that this novel representation of object dimensions in temporal correlations may be relevant to recent experiments on oscillatory neural firing in the visual cortex.

## 1   INTRODUCTION

Physical systems at a critical point exhibit long-range correlations even though the interactions among the constituent particles are of short range. Through the fluctuation-dissipation theorem this implies that the dynamics at one point in the

system are sensitive to external perturbations which are applied very far away. If we build an analog computer poised precisely at such a critical point it should be possible to evaluate highly non-local functionals of the input signals using a locally interconnected architecture. Such a scheme would be very useful for visual computations, especially those which require comparisons of widely separated regions of the image. From a biological point of view long-range correlations at a critical point might provide a robust scenario for "responses from beyond the classical receptive field" [1].

In this paper we present an explicit model for analog computation at a critical point and show that this model has a remarkable consequence: Because of dynamic scaling, spatial properties of input signals are mapped into temporal correlations of the local dynamics. One can, for example, measure the size and topology of "objects" in a scene using only the temporal correlations in the output of a single computational unit (neuron) located within the object. We then show that our abstract model can be realized in networks of semi-realistic spiking neurons. The key to this construction is that neurons biased in a regime of regular or oscillatory firing can be mapped to XY or planar spins [2,3], and two-dimensional arrays of these spins exhibit a broad range of parameters in which the system is generically at a critical point. Non-oscillatory neurons cannot, in general, be forced to operate at a critical point without delicate fine tuning of the dynamics, fine tuning which is implausible both for biology and for man-made analog circuits. We suggest that these arguments may be relevant to the recent observations of oscillatory firing in the visual cortex [4,5,6].

## 2  A STATISTICAL MECHANICS MODEL

We consider a simple two-dimensional array of spins whose states are defined by unit two-vectors $\mathbf{S}_n$. These spins interact with their neighbors so that the total energy of the system is $\mathbf{H} = -J \sum \mathbf{S}_n \cdot \mathbf{S}_m$, with the sum restricted to nearest neighbor pairs. This is the XY model, which is interesting in part because it possesses not a critical point but rather a critical *line* [7]. At a given temperature, for all $J > J_c$ one finds that correlations among spins decay algebraically, $\langle \mathbf{S}_n \cdot \mathbf{S}_m \rangle \propto 1/|\mathbf{r}_n - \mathbf{r}_m|^\eta$, so that there is no characteristic scale or correlation length; more precisely the correlation length is infinite. In contrast, for $J < J_c$ we have $\langle \mathbf{S}_n \cdot \mathbf{S}_m \rangle \propto \exp[-|\mathbf{r}_n - \mathbf{r}_m|/\xi]$, which defines a finite correlation length $\xi$.

In the algebraic phase the dynamics of the spins on long length scales are rigorously described by the spin wave approximation, in which one assumes that fluctuations in the angle between neighboring spins are small. In this regime it makes sense to use a continuum approximation rather than a lattice, and the energy of the system becomes $\mathbf{H} = J \int d^2r |\nabla \phi(\mathbf{x})|^2$, where $\phi(\mathbf{x})$ is the orientation of the spin at position $\mathbf{x}$. The dynamics of the system are determined by the Langevin equation

$$\frac{\partial \phi(\mathbf{x},t)}{\partial t} = J\nabla^2 \phi(\mathbf{x},t) + \eta(\mathbf{x},t), \tag{1}$$

where $\eta$ is a Gaussian thermal noise source with

$$\langle \eta(\mathbf{x},t)\eta(\mathbf{x}',t') \rangle = 2k_B T \delta(\mathbf{x} - \mathbf{x}')\delta(t - t'). \tag{2}$$

We can then show that the time correlation function of the spin at a single site $\mathbf{x}$ is given by

$$\langle \mathbf{S}(\mathbf{x},t) \cdot \mathbf{S}(\mathbf{x},0) \rangle = \exp\left[-2k_B T \int \frac{d\omega}{2\pi} \int \frac{d^2k}{(2\pi)^2} \frac{1-e^{-i\omega t}}{\omega^2 + J^2 k^4}\right]. \qquad (3)$$

In fact Eq. 3 is valid only for an infinite array of spins. Imagine that external signals to this array of spins can "activate" and "deactivate" the spins so that one must really solve Eq. 1 on finite regions or clusters of active spins. Then we can write the analog of Eq. 3 as

$$\langle \mathbf{S}(\mathbf{x},t) \cdot \mathbf{S}(\mathbf{x},0) \rangle = \exp\left[-\frac{k_B T}{J} \sum_n |\psi_n(\mathbf{x})|^2 \frac{1}{\lambda_n}(1 - e^{-J\lambda_n |t|})\right]. \qquad (4)$$

where $\psi_n$ and $\lambda_n$ are the eigenfunctions and associated eigenvalues of $(-\nabla^2)$ on the region of active spins. The key point here is that the spin auto-correlation function in time determines the spectrum of the Laplacian on the region of activity. But from the classic work of Kac [8] we know that this spectrum gives a great deal of information about the size and shape of the active region — we can in general determine the area, the length of the perimeter, and the topology (number of holes) from the set of eigenvalues $\{\lambda_n\}$, and this is true regardless of the absolute dimensions of the region. Thus by operating at a critical point we can achieve a scale-independent encoding of object dimension and topology in the temporal correlations of a locally connected system.

## 3    MAPPING REAL NEURONS ONTO THE STATISTICAL MODEL

All current models of neural networks are based on the hope that most microscopic ("biological") details are unimportant for the macroscopic, collective computational behavior of the system as a whole. Here we provide a rigorous connection between a more realistic neural model and a simplified model with spin variables and effective interactions, essentially the XY model discussed above. A more detailed account is given in [2,3].

We use the Fitzhugh-Nagumo (FN) model [9,10] to describe the electrical dynamics of an individual neuron. This model demonstrates a threshold for firing action potentials, a refractory period, and single-shot as well as repetitive firing — in short, all the qualitative properties of neural firing. It is also known to provide a reasonable quantitative description of several cell types. To be realistic it is essential to add a noise current $\delta I_n(t)$ which we take to be Gaussian, spectrally white, and independent in each cell $n$.

We connect each neuron to its neighbors in regular one- and two-dimensional arrays. More general local connections are easily added and do not significantly change the results presented below. We model a synapse between two neurons by exponentiating the voltage from one and injecting it as current into the other. Our choice is motivated by the fact that the number of transmitter vesicles released at a synapse is exponential in the presynaptic voltage [11]; other synaptic transfer characteristics, including small delays, give results qualitatively similar to those described

here. The resulting equations of motion are

$$\frac{dV_n}{dt} = (1/\tau_1) \left[ I_0 + \delta I_n(t) - V_n(V_n^2 - 1) - W_n + \sum_m J_{nm} \exp\{V_m(t)/V_0\} \right],$$

$$\frac{dW_n}{dt} = (1/\tau_2)[V_n - \alpha W_n], \tag{5}$$

where $V_n$ is the transmembrane voltage in cell $n$, $I_0$ is the DC bias current, and the $W_n$ are auxiliary variables; $V_0$ sets the scale of voltage sensitivity in the synapse. Voltages and currents are dimensionless, and the parameters of the system are expressed in terms of the time constants $\tau_1$ and $\tau_2$ and a dimensionless ratio $\alpha$. From the voltage traces we extract the spike arrival times in the $n^{th}$ neuron, $\{t_i^n\}$.

With the appropriate choice of parameters the FN model can be made to fire regularly—the interspike intervals are tightly clustered around a mean value. The power spectrum of the spike train $s(t) \sum_i \delta(t - t_i)$ has well resolved peaks at $\pm \omega_0$, $\pm 2\omega_0$, ... . We then low-pass filter $s(t)$ to keep only the $\pm \omega_0$ peaks, obtaining a phase-modulated cosine,

$$[Fs](t) \approx \omega_0 \cos[\omega_0 t + \phi(t)], \tag{6}$$

where $[Fs](t)$ denotes the filtered spike train. By looking at $[Fs](t)$ and its time derivative, we can extract the phase $\phi(t)$ which describes the oscillation that underlies regular firing. Since the orientation of a planar spin is also described by a single phase variable, we can reduce the spike train to a time-dependent planar spin $\mathbf{S}(t)$. We now want to see how these spins interact when we connect two cells via synapses.

We characterize the two-neuron interaction by accumulating a histogram of the phase differences between two connected neurons. This probability distribution defines an effective Hamiltonian, $P(\phi_1, \phi_2) \propto \exp[-\mathbf{H}(\phi_1 - \phi_2)]$. With excitatory synapses ($J > 0$) the interaction is ferromagnetic, as expected (see Fig. 1). The Hamiltonian takes other interesting forms for inhibitory, delayed, and nonreciprocal synapses. By simulating small clusters of cells we find that interactions other than nearest neighbor are negligible. This leads us to predict that the entire network is described by the effective Hamiltonian $\mathbf{H} = \sum_{ij} \mathbf{H}_{ij}(\phi_i - \phi_j)$, where $\mathbf{H}_{ij}(\phi_i - \phi_j)$ is the effective Hamiltonian measured for the pair of connected cells i, j.

One crucial consequence of Eq. 6 is that correlations of the filtered spike trains are exactly proportional to the spin-spin correlations which are natural objects in statistical mechanics. Specifically, if we have two cells $n$ and $m$,

$$\langle \mathbf{S}_n \cdot \mathbf{S}_m \rangle = \langle \cos(\phi_n - \phi_m) \rangle = \omega_0^{-2} \langle [Fs_n](t)[Fs_m](t) \rangle. \tag{7}$$

This relation shows us how the statistical description of the network can be tested in experiments which monitor actual neural spike trains.

## 4   DOES THE MAPPING WORK?

When planar spins are connected in a one-dimensional chain with nearest-neighbor interactions, correlations between spins drop off exponentially with distance. To test

this prediction we have run simulations on chains of 32 Fitzhugh-Nagumo neurons connected to their nearest neighbors. Correlations computed directly from the filtered spike trains as indicated above indeed decay exponentially, as seen in the insert to Fig. 1. Fig. 1 shows that the predictions for the correlation length from the simple model are in excellent agreement with the correlation lengths observed in the simulations of spiking neurons; there are no free parameters.

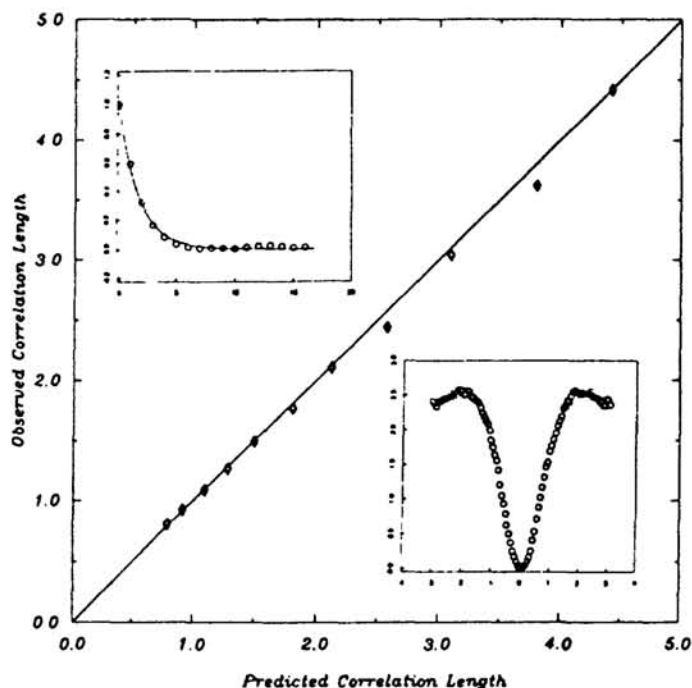

Figure 1: Correlation length obtained from fits to the simulation data vs. correlation length predicted from the Hamiltonians. Inset, upper left: Correlation function vs. distance from simulations, with exponential fit. Inset, lower right: Corresponding Hamiltonian as a function of phase difference.

In the two-dimensional case we connect each neuron to its four nearest neighbors on a square lattice. The corresponding spin model is essentially the XY mode. Hence we expect a low-temperature (high synaptic strength) phase with correlations that decay slowly (as a small power of distance) and a high-temperature (low synaptic strength) disordered phase with exponential decay. These predictions were confirmed by large-scale simulations of two-dimensional arrays [2].

## 5    OBJECT DIMENSIONS FROM TEMPORAL CORRELATIONS

We believe that we have presented convincing evidence for the description of regularly firing neurons in terms of XY spins, at least as regards their static or equilibrium correlations. In our theoretical discussion we showed that the temporal correlation functions of XY spins in the algebraic phase contained information about the

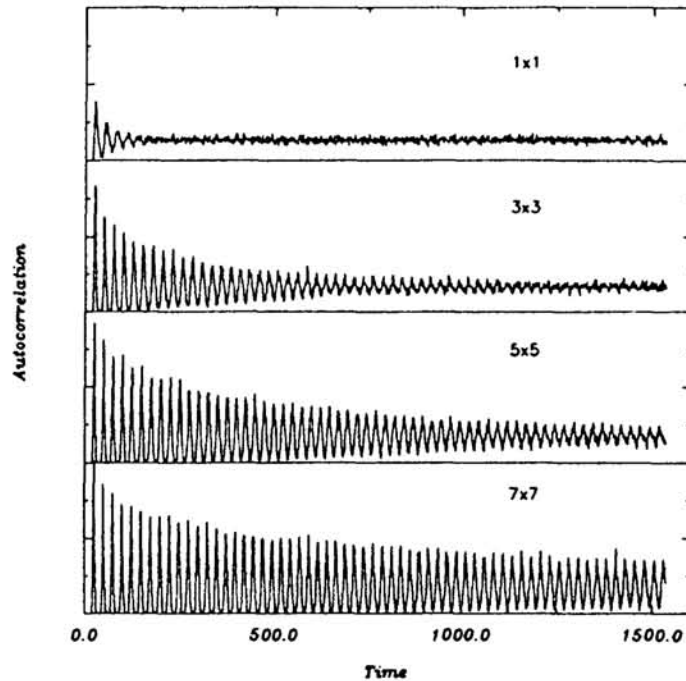

Figure 2: Auto-correlation functions for the spike trains of single cells at the center of square arrays of different sizes.

dimensions of "objects." Here we test this idea in a very simple numerical experiment. Imagine that we have an array of $N \times N$ connected cells which are excited by incoming signals so that they are in the oscillatory regime. Obviously we can measure the size of this "object" by looking at the entire network, but our theoretical results suggest that one can sense these dimensions ($N$) using the temporal correlations in just one cell, most simply the cell in the center of the array.

In Fig. 2 we show the auto-correlation functions for the spike trains of the center cell in arrays of different dimensions. It is clear that changing the dimensions of the array of active cells has profound effects on these spatially local temporal correlations. Because of the fact that the model is on a critical line these correlations continue to change as the dimensions of the array increase, rather than saturating after some finite correlation length is reached. Qualitatively similar results are expected throughout the algebraic phase of the associated spin model.

Recently it has been shown that when cells in the cat visual cortex are excited by appropriate stimuli they enter a regime of regular firing. These firing statistics are somewhat more complex than simulated here because there are a variable number of spikes per cycle, but we have reproduced all of our major results in models which capture this feature of the real data. We have seen that networks of regularly firing cells are capable of qualitatively different types of computation because these networks can be placed at a critical point without fine tuning of parameters. Most dramatically dynamic scaling allows us to trade spatial and temporal features and thereby encode object dimension in temporal correlations of single cells, as in Fig. 2. To see if such novel computations are indeed mediated by cortical oscillations

we suggest the direct analog of our numerical experiment, in which the correlation functions of single cells would be monitored in response to structured stimuli (e.g., textures) with different total spatial extent in the two dimensions of the visual field. We predict that these correlation functions will show a clear dependence on the area of the visual field being excited, with some sensitivity to the shape and topology as well. Most importantly this dependence on "object" dimension will extend to very large objects because the network is at a critical point. In this sense the temporal correlations of single cells will encode any object dimension, rather than being detectors for objects of some critical size.

## Acknowledgements

We thank O. Alvarez, D. Arovas, A. B. Bonds, K. Brueckner. M. Crair, E. Knobloch, H. Lecar, and D. Rohksar for helpful discussions. Work at Berkeley was supported in part by the National Science Foundation through a Presidential Young Investigator Award (to W.B.), supplemented by funds from Cray Research, Sun Microsystems, and the NEC Research Institute, by the Fannie and John Hertz Foundation through a Graduate Fellowship (to L.K.), and by the USPHS through a Biomedical Research Support Grant.

## Footnotes

*Current address.

## References

[1] J. Allman, F. Meizin, and E. McGuiness. *Ann. Rev. Neurosci.*, 8:407, 1985.

[2] L. Kruglyak. *From biological reality to simple physical models: Networks of oscillating neurons and the XY model.* PhD thesis, University of California at Berkeley, Berkeley, California, 1990.

[3] W. Bialek. In E. Jen, editor, *1989 Lectures in Complex Systems, SFI Studies in the Sciences of Complexity*, volume 2, pages 513–595. Addison-Wesley, Reading, Mass., 1990.

[4] R. Eckhorn, R. Bauer, W. Jordan, M. Brosch, W. Kruse, M. Munk, and H. J. Reitboeck. *Biol. Cybern.*, 60:121, 1988.

[5] C. M. Gray and W. Singer. *Proc. Nat. Acad. Sci. USA*, 86:1698, 1989.

[6] C. M. Gray, P. König, A. K. Engel, and W. Singer. *Nature*, 338:334, 1989.

[7] D. R. Nelson. In C. Domb and J. L. Lebowitz, editors, *Phase Transitions and Critical Phenomena*, volume 7, chapter 1. Academic Press, London, 1983.

[8] M. Kac. *The American Mathematical Monthly*, 73:1–23, 1966.

[9] Richard Fitzhugh. *Biophysical Journal*, 1:445–466, 1961.

[10] J. S. Nagumo, S. Arimoto, and S. Yoshizawa. *Proc. I. R. E.*, 50:2061, 1962.

[11] D. J. Aidley. *The Physiology of Excitable Cells.* Cambridge University Press, Cambridge, 1971.


